# Variational Inference for the Nested Chinese Restaurant Process

**Chong Wang**
Computer Science Department
Princeton University
chongw@cs.princeton.edu

**David M. Blei**
Computer Science Department
Princeton University
blei@cs.princeton.edu

## Abstract

The nested Chinese restaurant process (nCRP) is a powerful nonparametric Bayesian model for learning tree-based hierarchies from data. Since its posterior distribution is intractable, current inference methods have all relied on MCMC sampling. In this paper, we develop an alternative inference technique based on variational methods. To employ variational methods, we derive a tree-based stick-breaking construction of the nCRP mixture model, and a novel variational algorithm that efficiently explores a posterior over a large set of combinatorial structures. We demonstrate the use of this approach for text and hand written digits modeling, where we show we can adapt the nCRP to continuous data as well.

## 1 Introduction

For many application areas, such as text analysis and image analysis, learning a tree-based hierarchy is an appealing approach to illuminate the internal structure of the data. In such settings, however, the combinatoric space of tree structures makes model selection unusually daunting. Traditional techniques, such as cross-validation, require us to enumerate all possible model structures; this kind of methodology quickly becomes infeasible in the face of the set of all trees.

The *nested Chinese restaurant process* (nCRP) [1] addresses this problem by specifying a generative probabilistic model for tree structures. This model can then be used to discover structure from data using Bayesian posterior computation. The nCRP has been applied to several problems, such as fitting hierarchical topic models [1] and discovering taxonomies of images [2, 3].

The nCRP is based on the Chinese restaurant process (CRP) [4], which is closely linked to the Dirichlet process in its application to mixture models [5]. As a complicated Bayesian nonparametric model, posterior inference in an nCRP-based model is intractable, and previous approaches all rely Gibbs sampling [1, 2, 3]. While powerful and flexible, Gibbs sampling can be slow to converge and it is difficult to assess the convergence [6, 7]. Here, we develop an alternative for posterior inference for nCRP-based models.

Our solution is to use the optimization-based variational methods [8]. The idea behind variational methods is to posit a simple distribution over the latent variables, and then to fit this distribution to be close to the posterior of interest. Variational methods have been successfully applied to several Bayesian nonparametric models, such as Dirichlet process (DP) mixtures [9, 10, 11], hierarchical Dirichlet processes (HDP) [12], Pitman-Yor processes [13] and Indian buffet processes (IBP) [14].

The work presented here is unique in that our optimization of the variational distribution searches the combinatorial space of trees. Similar to Gibbs sampling, our method includes an exploration of a latent *structure* associated with the free parameters in addition to their values. First, we describe the tree-based stick-breaking construction of nCRP, which is needed for variational inference. Second, we develop our variational inference algorithm, which explores the infinite tree space associated with the nCRP. Finally, we study the performance of our algorithm on discrete and continuous data sets.

## 2 Nested Chinese restaurant process mixtures

The nested Chinese restaurant process (nCRP) is a distribution over hierarchical partitions [1]. It generalizes the Chinese restaurant process (CRP), which is a distribution over partitions. The CRP can be described by the following metaphor. Imagine a restaurant with an infinite number of tables, and imagine customers entering the restaurant in sequence. The $d$th customer sits at a table according to the following distribution,

$$p(c_d = k | c_{1:(d-1)}) \propto \begin{cases} m_k & \text{if } k \text{ is previous occupied} \\ \gamma & \text{if } k \text{ is a new table,} \end{cases} \qquad (1)$$

where $m_k$ is the number of previous customers sitting at table $k$ and $\gamma$ is a positive scalar. After $D$ customers have sat down, their seating plan describes a partition of $D$ items.

In the nested CRP, imagine now that tables are organized in a hierarchy: there is one table at the first level; it is associated with an infinite number of tables at the second level; each second-level table is associated with an infinite number of tables at the third level; and so on until the $L$th level. Each customer enters at the first level and comes out at the $L$th level, generating a path with $L$ tables as she sits in each restaurant. Moving from a table at level $\ell$ to one of its subtables at level $\ell + 1$, the customer draws following the CRP using Equation 1. (This description is slightly different from the metaphor in [1], but leads to the same distribution.)

The nCRP mixture model can be derived by analogy to the CRP mixture model [15]. (From now on, we will use the term "nodes" instead of "tables.") Each node is associated with a parameter $w$, where $w \sim G_0$ and $G_0$ is called the base distribution. Each data point is drawn by first choosing a path in the tree according to the nCRP, and then choosing its value from a distribution that depends on the parameters in that path. An additional hidden variable $x$ represents other latent quantities that can be used in this distribution. This is a generalization of the model described in [1]. For data $\mathcal{D} = \{t_n\}_{n=1}^N$, the nCRP mixture assumes that the $n$th data point $t_n$ is drawn as follows:

1. Draw a path $c_n | c_{1:(n-1)} \sim \text{nCRP}(\gamma, c_{1:(n-1)})$, which contains $L$ nodes from the tree.
2. Draw a latent variable $x_n \sim p(x_n | \lambda)$.
3. Draw an observation $t_n \sim p(t_n | W_{c_n}, x_n, \tau)$.

The parameters $\lambda$ and $\tau$ are associated with the latent variables $x$ and data generating distribution, respectively. Note that $W_{c_n}$ contains the $w_i$s selected by the path $c_n$. Specific applications of the nCRP mixture depend on the particular forms of $p(w)$, $p(x)$ and $p(t | W_c, x)$.

The corresponding posterior of the latent variables decomposes the data into a collection of paths, and provides distributions of the parameters attached to each node in those paths. Even though the nCRP assumes an "infinite" tree, the paths associated with the data will only populate a portion of that tree. Through this posterior, the nCRP mixture can be used as a flexible tree-based mixture model that does not assume a particular tree structure in advance of the data.

**Hierarchical topic models.** The nCRP mixture described above includes the hierarchical topic model of [1] as a special case. In that model, observed data are documents, i.e., a list of $N$ words from a fixed vocabulary. The nodes of the tree are associated with distributions over words ("topics"), and each document is associated with both a path in the tree and with a vector of proportions over its levels. Given a path, a document is generated by repeatedly generating level assignments from the proportions and then words from the corresponding topics. In the notation above, $p(w)$ is a Dirichlet distribution over the vocabulary simplex, $p(x)$ is a joint distribution of level proportions (from a Dirichlet) and level assignments ($N$ draws from the proportions), and $p(t | W_c, x)$ are the $N$ draws from the topics (for each word) associated with $x$.

**Tree-based hierarchical component analysis.** For continuous data, if $p(w)$, $p(x)$ and $p(t | W_c, x)$ are appropriate Gaussian distributions, we obtain hierarchical component analysis, a generalization of probabilistic principal component analysis (PPCA) [16, 17]. In this model, $w$ is the component parameter for the node it belongs to. Each path $c$ can be thought as a PPCA model with factor loading $W_c$ specified by that path. Then each data point chooses a path (also a PPCA model specified by that path) and draw the factors $x$. This model can also be thought as an infinite mixtures of PPCA model,

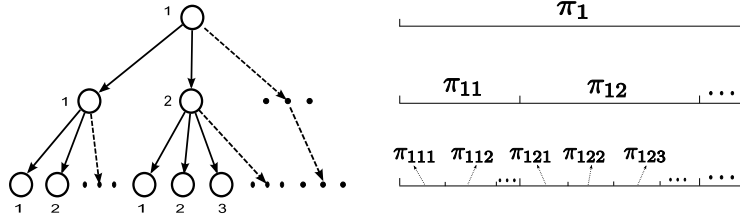

Figure 1: **Left.** A possible tree structure in a 3-level nCRP. **Right.** The tree-based stick-breaking construction of a 3-level nCRP.

where each PPCA can share components. In addition, we can incorporate the general exponential family PCA [18, 19] into the nCRP framework.[1]

### 2.1 Tree-based stick-breaking construction

CRP mixtures can be equivalently formulated using the Dirichlet process (DP) as a distribution over the distribution of each data point's random parameter [21, 4]. An advantage of expressing the CRP mixture with a DP is that the draw from the DP can be explicitly represented using the stick-breaking construction [22]. The DP bundles the scaling parameter $\gamma$ and base distribution $G_0$. A draw from a $DP(\gamma, G_0)$ is described as

$$v_i \sim \text{Beta}(1, \gamma), \quad \pi_i = v_i \prod_{j=1}^{i-1}(1 - v_j), \quad \boldsymbol{w}_i \sim G_0, \quad i \in \{1, 2, \cdots\}, \quad G = \sum_{i=1}^{\infty} \pi_i \delta_{\boldsymbol{w}_i},$$

where $\boldsymbol{\pi}$ are the stick lengths, and $\sum_{i=1}^{\infty} \pi_i = 1$ almost surely. This representation also illuminates the discreteness of a distribution drawn from a DP.

For the nCRP, we develop a similar stick-breaking construction. At the first level, the root node's stick length is $\pi_1 = v_1 \equiv 1$. For all the nodes at the second level, their stick lengths are constructed as for the DP, i.e., $\pi_{1i} = \pi_1 v_{1i} \prod_{j=1}^{i-1}(1 - v_{1j})$ for $i = \{1, 2, \cdots, \infty\}$ and $\sum_{i=1}^{\infty} \pi_{1i} = \pi_1 = 1$. The stick-breaking construction is then applied to each of these stick segments at the second level. For example, the $\pi_{11}$ portion of the stick is divided up into an infinite number of pieces according to the stick-breaking process. For the segment $\pi_{1k}$, the stick lengths of its children are $\pi_{1ki} = \pi_{1k} v_{1ki} \prod_{j=1}^{i-1}(1 - v_{1kj})$, for $i = \{1, 2, \cdots, \infty\}$ and $\sum_{i=1}^{\infty} \pi_{1ki} = \pi_{1k}$. The whole process continues for $L$ levels. This construction is best understood by Figure 1 (Right).

Although this stick represents an infinite tree, the nodes are countable and each node is uniquely identified by a sequence of $L$ numbers. We will denote all Beta draws as $\boldsymbol{V}$, each of which are independent draws from $\text{Beta}(1, \gamma)$ (except for the root $v_1$, which is equal to one).

The tree-based stick-breaking construction lets us calculate the conditional probability of a path given $\boldsymbol{V}$. Let the path $\boldsymbol{c} = [1, c_2, \cdots, c_L]$,

$$p(\boldsymbol{c}|\boldsymbol{V}) = \prod_{\ell=1}^{L} \pi_{1, c_2, \cdots, c_\ell} = \prod_{\ell=1}^{L} v_{1, c_2, \cdots, c_\ell} \prod_{j=1}^{c_\ell - 1}(1 - v_{1, c_2, \cdots, j}). \tag{2}$$

By integrating out $\boldsymbol{V}$ in Equation 2, we recover the nCRP. Given Equation 2, the joint probability of a data set under the nCRP mixture is

$$p(\boldsymbol{t}_{1:N}, \boldsymbol{x}_{1:N}, \boldsymbol{c}_{1:N}, \boldsymbol{V}, \boldsymbol{W}) = p(\boldsymbol{V})p(\boldsymbol{W}) \prod_{n=1}^{N} p(\boldsymbol{c}_n|\boldsymbol{V})p(\boldsymbol{x}_n)p(\boldsymbol{t}_n|\boldsymbol{W}_{\boldsymbol{c}_n}, \boldsymbol{x}_n). \tag{3}$$

This representation is the basis for variational inference.

## 3 Variational inference for the nCRP mixture

The central computational problem in Bayesian modeling is posterior inference: Given data, what is the conditional distribution of the latent variables in the model? In the nCRP mixture, these latent variables provide the tree structure and node parameters.

Posterior inference in an nCRP mixture has previously relied on Gibbs sampling, in which we sample from a Markov chain whose stationary distribution is the posterior [1, 2, 3]. Variational inference provides an alternative methodology: Posit a simple (e.g., factorized) family of distributions over the latent variables indexed by free parameters (called "variational parameters"). Then fit those parameters to be close in KL divergence to the true posterior of interest [8, 23].

Variational inference for Bayesian nonparametric models uses a truncated stick-breaking representation in the variational distribution [9] – free variational parameters are allowed only up to the truncation level. If the truncation is too large, the variational algorithm will still isolate only a subset of components; if the truncation is too small, methods have been developed to expand the truncated stick as part of the variational algorithm [10]. In the nCRP mixture, however, the challenge is that the tree structure is too large even to effectively truncate. We will address this by defining search criteria for adaptively adjusting the structure of the variational distribution, searching over the set of trees to best accommodate the data.

## 3.1 Variational inference based on the tree-based stick-breaking construction

We first address the problem of variational inference with a truncated tree of fixed structure. Suppose that we have a truncated tree $T$ and let $\mathcal{M}_T$ be the set of all nodes in $T$. Our family of variational distributions is defined as follows,

$$q(\boldsymbol{W}, \boldsymbol{V}, \boldsymbol{x}_{1:N}, \boldsymbol{c}_{1:N}) = \prod_{i \notin \mathcal{M}_T} q(\boldsymbol{w}_i)q(v_i) \prod_{i \in \mathcal{M}_T} p(\boldsymbol{w}_i)p(v_i) \prod_{n=1}^{N} q(\boldsymbol{c}_n)q(\boldsymbol{x}_n), \quad (4)$$

where: (1) Distributions $p(\boldsymbol{w}_i)$ and $p(v_i)$ for $i \notin \mathcal{M}_T$, are the prior distributions, containing *no* variational parameters; (2) Distributions $q(\boldsymbol{w}_i)$ and $q(v_i)$ for $i \in \mathcal{M}_T$ contain the variational parameters that we want to optimize for the truncated tree $T$; (3) Distribution $q(\boldsymbol{c}_n)$ is the variational multinomial distribution over *all* the possible paths, not just those in the truncated tree $T$. Note that there are *infinite* number of paths. We will address this issue below; (4) Distribution $q(\boldsymbol{x}_n)$ is the variational distribution for the latent variable $\boldsymbol{x}_n$ and it is in the same family of distribution, as $p(\boldsymbol{x}_n)$.

In summary, this family of distributions retains the infinite tree structure. Moreover, this family is nested [10, 11]: If a truncated tree $T_1$ is a subtree of a truncated tree $T_2$ then variational distributions defined over $T_1$ are a special case of those defined over $T_2$. Theoretically, the solution found using $T_2$ is at least as good as the one found using $T_1$. This allows us to use greedy search to find a better tree structure.

With the variational distributions (Equation 4) and the joint distributions (Equation 3), we turn to the details of posterior inference. Equivalent to minimizing KL is tightening the bound on the likelihood of the observations $\mathcal{D} = \{\boldsymbol{t}_n\}_{n=1}^{N}$ given by Jensen's inequality [8],

$$\log p(\boldsymbol{t}_{1:N}) \geq \mathbb{E}_q \left[ \log p(\boldsymbol{t}_{1:N}, \boldsymbol{V}, \boldsymbol{W}, \boldsymbol{x}_{1:N}, \boldsymbol{c}_{1:N}) \right] - \mathbb{E}_q \left[ \log q(\boldsymbol{V}, \boldsymbol{W}, \boldsymbol{x}_{1:N}, \boldsymbol{c}_{1:N}) \right]$$
$$= \sum_{i \in \mathcal{M}_T} \mathbb{E}_q \left[ \log \frac{p(\boldsymbol{w}_i)p(v_i)}{q(\boldsymbol{w}_i)q(v_i)} \right] + \sum_{n=1}^{N} \mathbb{E}_q \left[ \log \frac{p(\boldsymbol{x}_n)}{q(\boldsymbol{x}_n)} \right] + \sum_{n=1}^{N} \mathbb{E}_q \left[ \log \frac{p(\boldsymbol{t}_n|\boldsymbol{x}_n, \boldsymbol{W}_{\boldsymbol{c}_n})p(\boldsymbol{c}_n|\boldsymbol{V})}{q(\boldsymbol{c}_n)} \right]$$
$$\triangleq \mathcal{L}(q). \quad (5)$$

We optimize $\mathcal{L}(q)$ using coordinate ascent. First we isolate the terms that only contain $q(\boldsymbol{c}_n)$,

$$\mathcal{L}\left(q(\boldsymbol{c}_n)\right) = \mathbb{E}_q \left[ \log p(\boldsymbol{t}_n|\boldsymbol{x}_n, \boldsymbol{W}_{\boldsymbol{c}_n})p(\boldsymbol{c}_n|\boldsymbol{V}) \right] - \mathbb{E}_q \left[ \log q(\boldsymbol{c}_n) \right]. \quad (6)$$

Then we find the optimal solution for $q(\boldsymbol{c}_n)$ by setting the gradient to zero:

$$q(\boldsymbol{c}_n = \boldsymbol{c}) \propto S_{n,\boldsymbol{c}} \triangleq \exp \left\{ \mathbb{E}_q \left[ \log p(\boldsymbol{c}_n = \boldsymbol{c}|\boldsymbol{V}) \right] + \mathbb{E}_q \left[ \log p(\boldsymbol{t}_n|\boldsymbol{x}_n, \boldsymbol{W}_{\boldsymbol{c}}) \right] \right\}. \quad (7)$$

Since the values of $q(\boldsymbol{c}_n = \boldsymbol{c})$ is infinite, operating coordinate ascent over $q(\boldsymbol{c}_n = \boldsymbol{c})$ is difficult. We plug the optimal $q(\boldsymbol{c}_n)$ (Equation 7) into Equation 6 to obtain the lower bound

$$\mathcal{L}\left(q(\boldsymbol{c}_n)\right) = \log \sum_{\boldsymbol{c}} S_{n,\boldsymbol{c}}. \quad (8)$$

Two issues arise: 1) the variational distribution $q(\boldsymbol{c}_n)$ has infinite number of values, and we need to find an efficient way to manipulate this. 2) the lower bound $\log \sum_{\boldsymbol{c}} S_{n,\boldsymbol{c}}$ (Equation 8) contains infinite sum, which pose a problem in evaluation. In the appendix, we show that all the operations can be done only via the truncated tree $T$. We summarize the results as follows. Let $\bar{\boldsymbol{c}}$ be a path in $T$, either an *inner path* (a path ending at an inner node) or a *full path* (a path ending at a leaf node). Note that the inner path is only defined for the truncated tree $T$. The number of such $\bar{\boldsymbol{c}}$ is finite. In the

nCRP tree, denote child($\bar{c}$) as the set of all full paths that are *not* in $T$ but include $\bar{c}$ as a sub path. As a special case, if $\bar{c}$ is a full path, child($\bar{c}$) just contains itself. As shown in the appendix, we can compute these quantities efficiently:

$$q(\boldsymbol{c}_n = \bar{\boldsymbol{c}}) \triangleq \sum_{\boldsymbol{c}:\boldsymbol{c}\in\text{child}(\bar{\boldsymbol{c}})} q(\boldsymbol{c}_n = \boldsymbol{c}) \text{ and } S_{n,\bar{\boldsymbol{c}}} \triangleq \sum_{\boldsymbol{c}:\boldsymbol{c}\in\text{child}(\bar{\boldsymbol{c}})} S_{n,\boldsymbol{c}}. \tag{9}$$

Consequently iterating over the truncated tree $T$ using $\bar{c}$ is the same as iterating all the full paths in the nCRP tree. And these are all we need for doing variational inference.

Next, we move to optimize $q(v_i|a_i, b_i)$ for $i \in \mathcal{M}_T$, where $a_i$ and $b_i$ are variational parameters for Beta distribution $q(v_i)$. Let the path containing $v_i$ be $[1, c_2, \cdots, c_{\ell_0}]$, where $\ell_0 \leq L$. We isolate the term that only contains $v_i$ from the lower bound (Equation 5),

$$\mathcal{L}(q(v_i)) = \mathbb{E}_q \left[ \log p(v_i) - \log q(v_i) \right] + \sum_{n=1}^N \sum_{\boldsymbol{c}} q(\boldsymbol{c}_n = \boldsymbol{c}) \log p(\boldsymbol{c}_n = \boldsymbol{c}|\boldsymbol{V}). \tag{10}$$

After plugging Equation 2 into 10 and setting the gradient to be zero, we obtain the optimal $q(v_i)$,

$$\begin{aligned} q(v_i) &\propto v_i^{a_i^* - 1}(1 - v_i)^{b_i^* - 1}, \\ a_i^* &= 1 + \sum_{n=1}^N \sum_{c_{\ell_0+1}, \cdots, c_L} q(\boldsymbol{c}_n = [1, c_2, \cdots, c_{\ell_0}, c_{\ell_0+1}, \cdots, c_L]), \\ b_i^* &= \gamma + \sum_{n=1}^N \sum_{j, c_{\ell_0+1}, \cdots, c_L : j > c_{\ell_0}} q(\boldsymbol{c}_n = [1, c_2, \cdots, c_{\ell_0-1}, j, c_{\ell_0+1}, \cdots, c_L]), \end{aligned} \tag{11}$$

where the infinite sum involved can be solved using Equations 9.

The variational update functions for $\boldsymbol{W}$ and $\boldsymbol{x}$ depend on the actual distributions we use, and deriving them is straightforward. If they include an infinite sum then we apply similar techniques as we did for $q(v_i)$.

## 3.2 Refining the tree structure during variational inference

Since our variational distribution is nested, a larger truncated tree will always (theoretically) achieve a lower bound at least as tight as a smaller truncated tree. This allows us to search the infinite tree space until a certain criterion is satisfied (e.g., relative change of the lower bound). To achieve this, we present several heuristics to guide us to do so. All these operations are performed on the truncated tree $T$.

**Grow.** This operation is similar to what Gibbs sampling does in searching the tree space. We implement two heuristics: 1) Randomly choose several data points, and for each of them sample a path $\bar{c}$ according to $q(\boldsymbol{c}_n = \bar{\boldsymbol{c}})$. If it is an inner path, expand it a full path; 2) For every *inner* path in $T$, first compute the quantity $g(\bar{\boldsymbol{c}}) = \sum_{n=1}^N q(\boldsymbol{c}_n = \bar{\boldsymbol{c}})$. Then sample an inner path (say $\bar{\boldsymbol{c}}^*$) according to $g(\bar{\boldsymbol{c}})$, and expand it to full path.

**Prune.** If a certain path gets very little probability assignments from all data points, we eliminate this path – for path $\boldsymbol{c}$, the criterion is $\sum_{n=1}^N q(\boldsymbol{c}_n = \boldsymbol{c}) < \delta$, where $\delta$ is a small number. We use $\delta = 10^{-6}$). This mimics Gibbs sampling in the sense that for nCRP (or CRP), if a certain path (table) gets no assignments in the sampling process, it will never get any assignment any more according to Equation 1.

**Merge.** If paths $i$ and $j$ give almost equal posterior distributions, merging these two paths is employed [24]. The measure is $J(i, j) = \boldsymbol{P}_i^T \boldsymbol{P}_j / |\boldsymbol{P}_i||\boldsymbol{P}_j|$, where $\boldsymbol{P}_i = [q(\boldsymbol{c}_1 = i), \cdots, q(\boldsymbol{c}_N = i)]^T$. We use 0.95 as the threshold in our experiments.

In theory, *Prune* and *Merge* may decrease the lower bound. Empirically, we found even sometime it does, the effect is negligible. (but reduced the size of the tree). For continuous data settings, we additionally implement the *Split* method used in [24].

## 4 Experiments

In this section, we demonstrate variational inference for the nCRP. We analyze both discrete and continuous data using the two applications discussed in Section 2.

| Per-word test set likelihood | | | |
|---|---|---|---|
| Method | JACM | Psy. Review | PNAS |
| Gibbs sampling | $-5.3922 \pm 0.0052$ | $-5.7834 \pm 0.0149$ | $-6.4961 \pm 0.0068$ |
| Var. inference | $-5.4331 \pm 0.0100$ | $-5.8430 \pm 0.0153$ | $-6.5736 \pm 0.0050$ |
| Var. inference (G) | $-5.4495 \pm 0.0118$ | $-5.8593 \pm 0.0157$ | $-6.5996 \pm 0.0153$ |

Table 1: Test set likelihood comparison on three datasets. Var. inference (G): variational inference initialized from the initialization of Gibbs sampling. Variational inference can give competitive performance on test set likelihood.

## 4.1 Hierarchical topic modeling

For discrete data, we compare variational inference compared with Gibbs sampling for hierarchical topic modeling. Three corpora are used in the experiments: (1) **JACM**: a collection of 536 abstracts from the *Journal of the ACM* from years 1987 to 2004 with a vocabulary size of 1,539 and around 68K words; (2) **Psy. Review**: a collection of 1,272 psychology abstracts from *Psychological Review* from years 1967 to 2003, with a vocabulary size of 1,971 and around 137K words; (3) **PNAS**: a collection of 5,000 abstracts from the *Proceedings of the National Academy of Sciences* from years 1991 to 2001, with a vocabulary size of 7762 and around 895K words. Those terms occurring in fewer than 5 documents were removed.

Local maxima can be a problem for both Gibbs sampling and variational inference. To avoid them in Gibbs sampling, we randomly restart the sampler 200 times and take the trajectory with the highest average posterior likelihood. We run the Gibbs sampling for 10000 iterations and collect the results for post analysis. For variational inference, we use two types of initializations 1) similar to Gibbs sampling, we gradually add data points during the variational inference as well – add a new path for each document in the initialization; 2) we initialize the variational inference from the initialization for Gibbs sampling – using the MAP estimate using one Gibbs sample. We set $L = 3$ for all the experiments and use the same hyperparameters in both algorithms. Specifically, the stick-breaking prior parameter $\gamma$ is set to 1.0; the symmetric Dirichlet prior parameter for the topics is set to 1.0; the prior for level proportions is skewed to favor high levels $(50, 20, 10)$. (This is suggested in [1].) We run the variational inference until the relative change of log-likelihood is less than 0.001.

**Per-word test set likelihood.** We use test set likelihood as a measure of performance. The procedure is to divide the corpus into a training set $D_{\text{train}}$ and a test set $D_{\text{test}}$, and approximate the likelihood of $D_{\text{test}}$ given $D_{\text{train}}$. We use the same method in Teh et al. [12] to approximate it. Specifically, we use posterior means $\hat{\theta}$ and $\hat{\beta}$ to represent the estimated topic mixture proportions over $L$ levels and topic multinomial parameters. For the variational method, we use

$$p(\{\boldsymbol{t}_1, \cdots, \boldsymbol{t}_N\}_{\text{test}}) = \prod_{n=1}^{N} \sum_{\boldsymbol{c}} q(\boldsymbol{c}_n = \boldsymbol{c}) \prod_j \sum_{n,\ell} \hat{\theta}_{n,\ell} \hat{\beta}_{c_\ell, t_{nj}},$$

where $\hat{\theta}$ and $\hat{\beta}$ are estimated using mean values from the variational distributions. For Gibbs sampling, we use $S$ samples and compute

$$p(\{\boldsymbol{t}_1, \cdots, \boldsymbol{t}_N\}_{\text{test}}) = \prod_{n=1}^{N} \frac{1}{S} \sum_{s=1}^{S} \sum_{\boldsymbol{c}} \delta_{\boldsymbol{c}_n^s} \prod_j \sum_{n,\ell} \hat{\theta}_{n,\ell}^s \hat{\beta}_{c_\ell, t_{nj}}^s,$$

where $\hat{\theta}^s$ and $\hat{\beta}^s$ are estimated using sample $s$ [25, 12]. We use 30 samples collected at a lag of 10 after a 200-sample burn-in for a document in test set. Actually, $1/S \sum_{s=1}^{S} \sum_{\boldsymbol{c}} \delta_{\boldsymbol{c}_n^s}$ gives the empirical estimation of $p(\boldsymbol{c}_n)$, where in variational inference, we approximate it using $q(\boldsymbol{c}_n)$. Table 1 shows the test likelihood comparison using five-fold cross validation. This shows our model can give competitive performance in term of the test set likelihood. This discrepancy is similar to that in [12] when variational inference is compared the collapsed Gibbs sampling for HDP.

**Topic visualizations.** Figures 2 and 3 show the tree-based topic visualizations from JACM and PNAS datasets. These are quite similar to those obtained by Gibbs sampling (see [1]).

## 4.2 Modeling handwritten digits using hierarchical component analysis

For continuous data, we use the hierarchical component analysis for modeling handwritten digits (http://archive.ics.uci.edu/ml). This dataset contains 3823 handwritten digits as a training set and

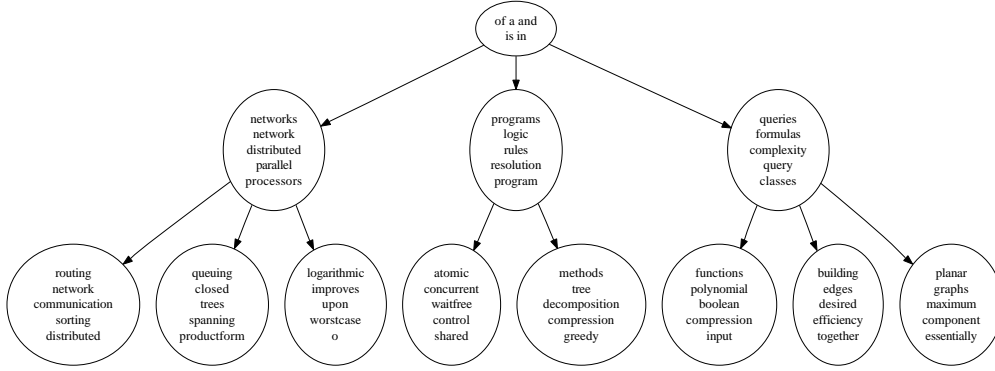

Figure 2: A sub network discovered on JACM dataset, each topic represented by top 5 terms. The whole tree has 30 nodes, with an average branching factor 2.64.

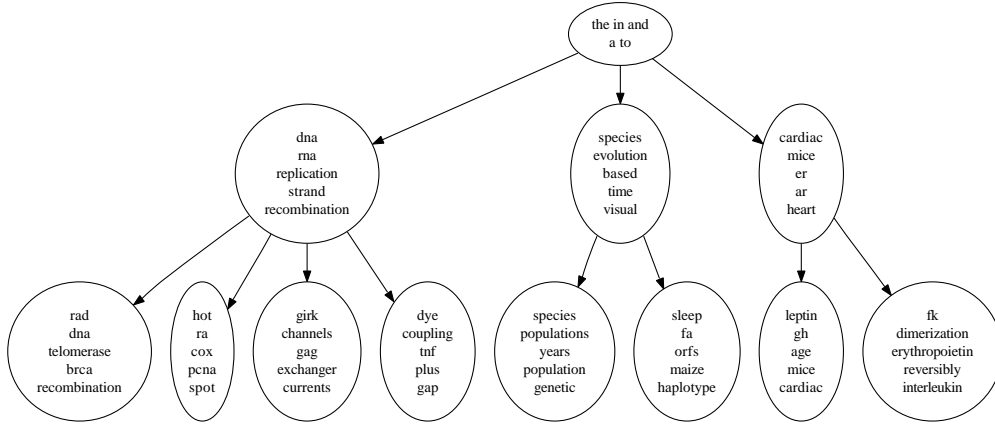

Figure 3: A sub network discovered on PNAS dataset, each topic represented by top 5 terms. The whole tree has 45 nodes, with an average branching factor 2.93.

1797 as a testing set. Each digit contains 64 integer attributes, ranging from 0-16. As described in section 2, we use PPCA [16] as the basic model for each path. We use a global mean parameter $\boldsymbol{\mu}$ for all paths, although a model with an individual mean parameter for each path can be similarly derived. We put broad priors over the parameters similar to those in variational Bayesian PCA [17]. The stick-breaking prior parameter $\gamma = 1$ is set to be 1.0; for each node, $\boldsymbol{w} \sim \mathcal{N}(\boldsymbol{0}, 10^3)$; $\boldsymbol{\mu} \sim \mathcal{N}(\boldsymbol{0}, 10^3)$; the inverse of the variance for the noise model in PPCA is $\tau$ and $\tau \sim \mathrm{Gamma}(10^{-3}, 10^{-3})$. Again, we run the variational inference until the relative change of log-likelihood is less than 0.001.

We compare the reconstruction error with PCA. To compute the reconstruction error for our model, we first select the path for each data point using its MAP estimation by $\hat{c}_n = \arg\max_{\boldsymbol{c}} q(\boldsymbol{c}_n = c)$. Then we use the similar approach [26, 24] to reconstruct $\boldsymbol{t}_n$,

$$\hat{\boldsymbol{t}}_n = \boldsymbol{W}_{\hat{\boldsymbol{c}}_n}(\boldsymbol{W}_{\hat{\boldsymbol{c}}_n}{}^T\boldsymbol{W}_{\hat{\boldsymbol{c}}_n})^{-1}\boldsymbol{W}_{\hat{\boldsymbol{c}}_n}{}^T(\boldsymbol{t}_n - \hat{\boldsymbol{\mu}}) + \hat{\boldsymbol{\mu}}.$$

We test our model using depth $L = 2, 3, 4, 5$. All of our models run within 2 minutes. The reconstruction errors for both the training and testing set are shown in Table 2. Our model gives lower reconstruction errors than PCA.

## 5 Conclusions

In this paper, we presented the variational inference algorithm for the nested Chinese restaurant process based on its tree-based stick-breaking construction. Our result indicates that the variational

| Reconstruction error on handwritten digits | | | | |
|---|---|---|---|---|
| #Depth | HCA (tr) | PCA (tr) | HCA (te) | PCA (te) |
| 2(9) | **631.6** | 863.0 | **699.4** | 878.5 |
| 3(14) | **559.8** | 722.3 | **585.6** | 727.7 |
| 4(18) | **463.4** | 621.0 | **506.1** | 633.0 |
| 5(22) | **384.8** | 553.0 | **461.8** | 564.2 |

Table 2: Reconstruction error comparison (Tr: train; Te: test). HCA stands for hierarchical component analysis. PCA uses $L$ largest components. In the first column, 2(9) means $L = 2$ with 9 nodes inferred using our model. Others are similarly defined. HCA gives lower reconstruction errors.

inference is a powerful alternative method for the widely used Gibbs sampling. We also adapt the nCRP to model continuous data, e.g. in hierarchical component analysis.

**Acknowledgements.** We thank anonymous reviewers for insightful suggestions. David M. Blei is supported by ONR 175-6343, NSF CAREER 0745520, and grants from Google and Microsoft.

## Appendix: efficiently manipulating $S_{n,\boldsymbol{c}}$ and $q(\boldsymbol{c}_n = \boldsymbol{c})$

**Case 1: All nodes of the path are in $T$, $\boldsymbol{c} \subset \mathcal{M}_T$.** Let $Z_0 \triangleq \mathbb{E}_q \left[ \log p(\boldsymbol{t}_n | \boldsymbol{x}_n, \boldsymbol{W_c}) \right]$. We have

$$ S_{n,\boldsymbol{c}} = \exp \left\{ \mathbb{E}_q \left[ \sum_{\ell=1}^L (\log(v_{1,c_2,\cdots,c_\ell}) + \sum_{j=1}^{c_\ell-1} \log(1 - v_{1,c_2,\cdots,j})) \right] + Z_0 \right\}. \tag{12} $$

**Case 2: At least one node is not in $T$, $\boldsymbol{c} \not\subset \mathcal{M}_T$.** Although $\boldsymbol{c} \not\subset \mathcal{M}_T$, $\boldsymbol{c}$ must have some nodes in $\mathcal{M}_T$. Then $\boldsymbol{c}$ can be written as $\boldsymbol{c} = [\bar{\boldsymbol{c}}, c_{\ell_0+1}, \cdots, c_L]$, where $\bar{\boldsymbol{c}} \triangleq [1, c_2, \cdots, c_{\ell_0}] \subset \mathcal{M}_T$ and $[\bar{\boldsymbol{c}}, c_{\ell_0+1}, \cdots, c_\ell] \not\subset \mathcal{M}_T$ for any $\ell > \ell_0$. In the truncated tree $T$, let $j_0$ be the maximum index for the child node whose parent path is $\bar{\boldsymbol{c}}$, then we know if $c_{\ell_0+1} > j_0$, $[\bar{\boldsymbol{c}}, c_{\ell_0+1}, \cdots, c_L] \not\subset \mathcal{M}_T$. Now we fix the sub path $\bar{\boldsymbol{c}}$ and let $[c_{\ell_0+1}, \cdots, c_L]$ vary (satisfying $c_{\ell_0+1} > j_0$). All these possible paths constitute a set: $\text{child}(\bar{\boldsymbol{c}}) \triangleq \{[\bar{\boldsymbol{c}}, c_{\ell_0+1}, \cdots, c_L] : c_{\ell_0+1} > j_0\}$. According to Equation 4, for any $\boldsymbol{c} \in \text{child}(\bar{\boldsymbol{c}})$, $Z_0 \triangleq \mathbb{E}_q \left[ \log p(\boldsymbol{t}_n | \boldsymbol{x}_n, \boldsymbol{W_c}) \right]$ is constant, since the variational distribution for $\boldsymbol{w}$ outside the truncated tree is the same prior distribution. We have

$$ \sum_{\boldsymbol{c} \in \text{child}(\bar{\boldsymbol{c}})} S_{n,\boldsymbol{c}} $$

$$ = \sum_{\boldsymbol{c} \in \text{child}(\bar{\boldsymbol{c}})} \exp \left\{ Z_0 + \mathbb{E}_q \left[ \sum_{\ell=1}^L (\log(v_{1,\cdots,c_\ell}) + \sum_{j=1}^{c_\ell-1} \log(1 - v_{1,c_2,\cdots,j})) \right] \right\} $$

$$ = \frac{\exp(Z_0 + (L-\ell_0)\mathbb{E}_p[\log(v)])}{(1 - \exp(\mathbb{E}_p[\log(1-v)]))^{L-\ell_0}} \exp \left\{ \mathbb{E}_q \left[ \sum_{\ell=1}^{\ell_0} (\log(v_{1,c_2,\cdots,c_\ell}) + \sum_{j=1}^{c_\ell-1} \log(1 - v_{1,c_2,\cdots,j})) \right] \right\} $$

$$ \times \exp \left( \mathbb{E}_q \left[ \sum_{j=1}^{j_0} \log(1 - v_{1,c_2,\cdots,c_{\ell_0},j}) \right] \right), \tag{13} $$

where $v \sim \text{Beta}(1, \gamma)$. Such cases contain all inner nodes in the truncated tree $T$. Note that Case 1 is a special case of Case 2 by setting $\ell_0 = L$. Given all these, $\sum_{\boldsymbol{c}} S_{n,\boldsymbol{c}}$ can be computed efficiently. Furthermore, given Equations 13 and Equation 7, we define

$$ q(\boldsymbol{c}_n = \bar{\boldsymbol{c}}) \triangleq \sum_{\boldsymbol{c} \in \text{child}(\bar{\boldsymbol{c}})} q(\boldsymbol{c}_n = \boldsymbol{c}) \propto \sum_{\boldsymbol{c} \in \text{child}(\bar{\boldsymbol{c}})} S_{n,\boldsymbol{c}}, \tag{14} $$

which corresponds the sum of probabilities from all paths in $\text{child}(\bar{\boldsymbol{c}})$. We note that this organization only depends on the truncated tree $T$ and is sufficient for variational inference.

## Footnotes

[1]We note that Bach and Jordan [20] studied tree-dependent component analysis, a generalization of independent component analysis where the components are organized in a tree. This model expresses a different philosophy: Their tree reflects the actual conditional dependencies among the components. Data are not generated by choosing a path first, but by a linear transformation of all components in the tree.

# References

[1] Blei, D. M., T. L. Griffiths, M. I. Jordan, et al. Hierarchical topic models and the nested Chinese restaurant process. In *NIPS*. 2003.

[2] Bart, E., I. Porteous, P. Perona, et al. Unsupervised learning of visual taxonomies. In *CVPR*. 2008.

[3] Sivic, J., B. C. Russell, A. Zisserman, et al. Unsupervised discovery of visual object class hierarchies. In *CVPR*. 2008.

[4] Aldous, D. Exchangeability and related topics. In *Ecole d'Ete de Probabilities de Saint-Flour XIII 1983*, pages 1–198. Springer, 1985.

[5] Ferguson, T. S. A Bayesian analysis of some nonparametric problems. *The Annals of Statistics*, 1(2):209–230, 1973.

[6] Neal, R. Probabilistic inference using Markov chain Monte Carlo methods. Tech. Rep. CRG-TR-93-1, Department of Computer Science, University of Toronto, 1993.

[7] Robert, C., G. Casella. *Monte Carlo Statistical Methods*. Springer-Verlag, New York, NY, 2004.

[8] Jordan, M. I., Z. Ghahramani, T. S. Jaakkola, et al. An introduction to variational methods for graphical models. *Learning in Graphical Models*, 1999.

[9] Blei, D. M., M. I. Jordan. Variational methods for the Dirichlet process. In *ICML*. 2004.

[10] Kurihara, K., M. Welling, N. A. Vlassis. Accelerated variational Dirichlet process mixtures. In *NIPS*. 2006.

[11] Kurihara, K., M. Welling, Y. W. Teh. Collapsed variational Dirichlet process mixture models. In *IJCAI*. 2007.

[12] Teh, Y. W., K. Kurihara, M. Welling. Collapsed variational inference for HDP. In *NIPS*. 2008.

[13] Sudderth, E. B., M. I. Jordan. Shared segmentation of natural scenes using dependent Pitman-Yor processes. In *NIPS*. 2008.

[14] Doshi, F., K. T. Miller, J. Van Gael, et al. Variational inference for the Indian buffet process. In *AISTATS*, vol. 12. 2009.

[15] Escobar, M. D., M. West. Bayesian density estimation and inference using mixtures. *Journal of the American Statistical Association*, 90:577–588, 1995.

[16] Tipping, M. E., C. M. Bishop. Probabilistic principal component analysis. *Journal of the Royal Statistical Society, Series B*, 61:611–622, 1999.

[17] Bishop, C. M. Variational principal components. In *ICANN*. 1999.

[18] Collins, M., S. Dasgupta, R. E. Schapire. A generalization of principal components analysis to the exponential family. In *NIPS*. 2001.

[19] Mohamed, S., K. A. Heller, Z. Ghahramani. Bayesian exponential family PCA. In *NIPS*. 2008.

[20] Bach, F. R., M. I. Jordan. Beyond independent components: Trees and clusters. *JMLR*, 4:1205–1233, 2003.

[21] Antoniak, C. E. Mixtures of Dirichlet processes with applications to Bayesian nonparametric problems. *The Annals of Statistics*, 2(6):1152–1174, 1974.

[22] Sethuraman, J. A constructive definition of Dirichlet priors. *Statistica Sinica*, 4:639–650, 1994.

[23] Wainwright, M., M. Jordan. Variational inference in graphical models: The view from the marginal polytope. In *Allerton Conference on Control, Communication and Computation*. 2003.

[24] Ueda, N., R. Nakano, Z. Ghahramani, et al. SMEM algorithm for mixture models. *Neural Computation*, 12(9):2109–2128, 2000.

[25] Griffiths, T. L., M. Steyvers. Finding scientific topics. *Proc Natl Acad Sci USA*, 101 Suppl 1:5228–5235, 2004.

[26] Tipping, M. E., C. M. Bishop. Mixtures of probabilistic principal component analysers. *Neural Computation*, 11(2):443–482, 1999.

